# Object Bank: A High-Level Image Representation for Scene Classification & Semantic Feature Sparsification

**Li-Jia Li\*[1], Hao Su\*[1], Eric P. Xing[2], Li Fei-Fei[1]**
1 Computer Science Department, Stanford University
2 Machine Learning Department, Carnegie Mellon University

## Abstract

Robust low-level image features have been proven to be effective representations for a variety of visual recognition tasks such as object recognition and scene classification; but pixels, or even local image patches, carry little semantic meanings. For high level visual tasks, such low-level image representations are potentially not enough. In this paper, we propose a high-level image representation, called the Object Bank, where an image is represented as a scale-invariant response map of a large number of pre-trained generic object detectors, blind to the testing dataset or visual task. Leveraging on the Object Bank representation, superior performances on high level visual recognition tasks can be achieved with simple off-the-shelf classifiers such as logistic regression and linear SVM. Sparsity algorithms make our representation more efficient and scalable for large scene datasets, and reveal semantically meaningful feature patterns.

## 1 Introduction

Understanding the meanings and contents of images remains one of the most challenging problems in machine intelligence and statistical learning. Contrast to inference tasks in other domains, such as NLP, where the basic feature space in which the data lie usually bears explicit human perceivable meaning, e.g., each dimension of a document embedding space could correspond to a word [21], or a topic, common representations of visual data seem to primarily build on raw physical metrics of the pixels such as color and intensity, or their mathematical transformations such as various filters, or simple image statistics such as shape, edges orientations etc. Depending on the specific visual inference task, such as classification, a predictive method is deployed to pool together and model the statistics of the image features, and make use of them to build some hypothesis for the predictor. For example, Fig.1 illustrates the gradient-based GIST features [25] and texture-based Spatial Pyramid representation [19] of two different scenes (foresty mountain vs. street). But such schemes often fail to offer sufficient discriminative power, as one can see from the very similar image statistics in the examples in Fig.1.

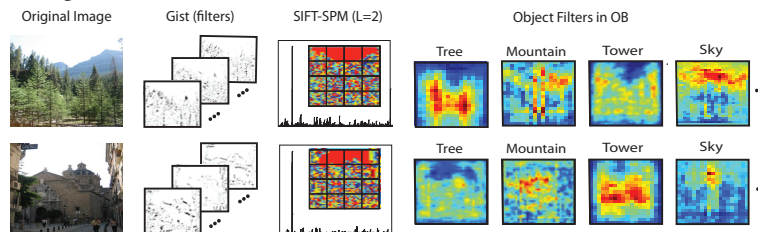

Figure 1: (Best viewed in colors and magnification.) Comparison of object bank (OB) representation with two low-level feature representations, GIST and SIFT-SPM of two types of images, mountain vs. city street. From left to right, for each input image, we show the selected filter responses in the GIST representation [25], a histogram of the SPM representation of SIFT patches [19], and a selected number of OB responses.

While more sophisticated low-level feature engineering and recognition model design remain important sources of future developments, we argue that the use of semantically more meaningful feature space, such as one that is directly based on the content (e.g., objects) of the images, as words for textual documents, may offer another promising venue to empower a computational visual recognizer to potentially handle arbitrary natural images, especially in our current era where visual knowledge of millions of common objects are readily available from various easy sources on the Internet.

In this paper, we propose "Object Bank" (OB), a new representation of natural images based on objects, or more rigorously, a collection of object sensing filters built on a generic collection of labeled objects. We explore how a simple linear hypothesis classifier, combined with a sparse-coding scheme, can leverage on this representation, despite its extreme high-dimensionality, to achieve superior predictive power over similar linear prediction models trained on conventional representations. We show that an image representation based on objects can be very useful in high-level visual recognition tasks for scenes cluttered with objects. It provides complementary information to that of the low-level features. As illustrated in Fig.1, these two different scenes show very different image responses to objects such as tree, street, water, sky, etc. Given the availability of large-scale image datasets such as LabelMe [30] and ImageNet [5], it is no longer inconceivable to obtain trained object detectors for a large number of visual concepts. In fact we envision the usage of thousands if not millions of these available object detectors as the building block of such image representation in the future.

While the OB representation offers a rich, high-level description of images, a key technical challenge due to this representation is the "curse of dimensionality", which is severe because of the size (i.e., number of objects) of the object bank and the dimensionality of the response vector for each object. Typically, for a modest sized picture, even hundreds of object detectors would result into a representation of tens of thousands of dimensions. Therefore to achieve robust predictor on practical dataset with typically only dozens or a couple of hundreds of instances per class, structural risk minimization via appropriate regularization of the predictive model is essential.

In this paper, we propose a regularized logistic regression method, akin to the group lasso approach for structured sparsity, to explore both *feature sparsity* and *object sparsity* in the Object Bank representation for learning and classifying complex scenes. We show that by using this high-level image representation and a simple sparse coding regularization, our algorithm not only achieves superior image classification results in a number of challenging scene datasets, but also can discover semantically meaningful descriptions of the learned scene classes.

## 2 Related Work

A plethora of image descriptors have been developed for object recognition and image classification [25, 1, 23]. We particularly draw the analogy between our *object* bank and the *texture* filter banks [26, 10].

Object detection and recognition also entail a large body of literature [7]. In this work, we mainly use the current state-of-the-art object detectors of Felzenszwalb et. al. [9], as well as the geometric context classifiers ("stuff" detectors) of Hoeim et. al. [13] for pre-training the object detectors.

The idea of using object detectors as the basic representation of images is analogous [12, 33, 35]. In contrast to our work, in [12] and [33] each semantic concept is trained by using the entire images or frames of video. As there is no localization of object concepts in scenes, understanding cluttered images composed of many objects will be challenging. In [35], a small number of concepts are trained and only the most probable concept is used to form the representation for each region, whereas in our approach all the detector responses are used to encode richer semantic information.

The idea of using many object detectors as the basic representation of images is analogous to approaches applying a large number of "semantic concepts" to video and image annotation and retrieval [12, 33, 35]. In contrast to our work, in [12, 33, 35] each semantic concept is trained by using entire images or frames of videos. There is no sense of localized representation of meaningful object concepts in scenes. As a result, this approach is difficult to use for understanding cluttered images composed of many objects.

Combinations of small set of ($\sim$ a dozen of) off-the-shelf object detectors with global scene context have been used to improve object detection [14, 28, 29]. Also related to our work is a very recent exploration of using attributes for recognition [17, 8, 16]. But we emphasize such usage is not a

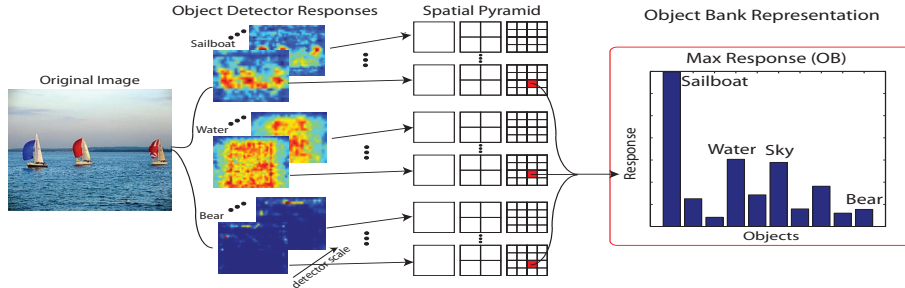

Figure 2: (Best viewed in colors and magnification.) Illustration of *OB*. A large number of object detectors are first applied to an input image at multiple scales. For each object at each scale, a three-level spatial pyramid representation of the resulting object filter map is used, resulting in $\texttt{No.Objects} \times \texttt{No.Scales} \times (1^2 + 2^2 + 4^2)$ grids; the maximum response for each object in each grid is then computed, resulting in a $\texttt{No.Objects}$ length feature vector for each grid. A concatenation of features in all grids leads to an OB descriptor for the image.

universal representation of images as we have proposed. To our knowledge, this is the first work that use such high-level image features at different image location and scale.

## 3 The Object Bank Representation of Images

*Object Bank* (OB) is an image representation constructed from the responses of many object detectors, which can be viewed as the response of a "generalized object convolution." We use two state-of-the-art detectors for this operation: the latent SVM object detectors [9] for most of the blobby objects such as tables, cars, humans, etc, and a texture classifier by Hoiem [13] for more texture- and material-based objects such as sky, road, sand, etc. We point out here that we use the word "object" in its very general form – while cars and dogs are objects, so are sky and water. Our image representation is agnostic to any specific type of object detector; we take the "outsourcing" approach and assume the availability of these pre-trained detectors.

Fig. 2 illustrates the general setup for obtaining the OB representation. A large number of object detectors are run across an image at different scales. For each scale and each detector, we obtain an initial response map of the image (see Appendix for more details of using the object detectors [9, 13]). In this paper, we use 200 object detectors at 12 detection scales and 3 spatial pyramid levels (L=0,1,2) [19]. We note that this is a universal representation of any images for any tasks. We use the same set of object detectors regardless of the scenes or the testing dataset.

### 3.1 Implementation Details of Object Bank

So what are the "objects" to use in the object bank? And how many? An obvious answer to this question is to use all objects. As the detectors become more robust, especially with the emergence of large-scale datasets such as LabelMe [30] and ImageNet [5], this goal becomes more reachable.

But time is not fully ripe yet to consider using all objects in, say, the LabelMe dataset. Not enough research has yet gone into building robust object detector for tens of thousands of generic objects. And even more importantly, not all objects are of equal importance and prominence in natural images. As Fig.1 in Appendix shows, the distribution of objects follows Zipf's Law, which implies that a small proportion of object classes account for the majority of object instances.

For this paper, we will choose a few hundred most useful (or popular) objects in images[1]. An important practical consideration for our study is to ensure the availability of enough training images for each object detectors. We therefore focus our attention on obtaining the objects from popular image datasets such as ESP [31], LabelMe [30], ImageNet [5] and the Flickr online photo sharing community. After ranking the objects according to their frequencies in each of these datasets, we take the intersection set of the most frequent 1000 objects, resulting in 200 objects, where the identities and semantic relations of some of them are illustrated in Fig.2 in the Appendix. To train each of the 200 object detectors, we use 100~200 images and their object bounding box information from the LabelMe [30] (86 objects) and ImageNet [5] datasets (177 objects). We use a subset of LabelMe scene dataset to evaluate the object detector performance. Final object detectors are selected based on their performance on the validation set from LabelMe (see Appendix for more details).

# 4 Scene Classification and Feature/Object Compression via Structured Regularized Learning

We envisage that with the avalanche of annotated objects on the web, the number of object detectors in our object bank will increase quickly from hundreds to thousands or even millions, offering increasingly rich signatures for each images based on the identity, location, and scale of the object-based content of the scene. However, from a learning point of view, it also poses a challenge on how to train predictive models built on such high-dimensional representation with limited number of examples. We argue that, with an "overcomplete" OB representation, it is possible to compress ultra-high dimensional image vector without losing semantic saliency. We refer this semantic-preserving compression as *content-based compression* to contrast the conventional information-theoretic compression that aims at lossless reconstruction of the data.

In this paper, we intend to explore the power of OB representation in the context of Scene Classification, and we are also interested in discovering meaningful (possibly small subset of) dimensions during regularized learning for different classes of scenes. For simplicity, here we present our model in the context of linear binary classier in a 1-versus-all classification scheme for $K$ classes. Generalization to a multiway softmax classifier is slightly more involved under structured regularization and thus deferred to future work. Let $\mathbf{X} = [\mathbf{x}_1^T; \mathbf{x}_2^T; \ldots; \mathbf{x}_N^T] \in \mathbb{R}^{N \times J}$, an $N \times J$ matrix, represent the design built on the $J$-dimensional object bank representation of $N$ images; and let $\mathbf{Y} = (y_1, \ldots, y_N) \in \{0, 1\}^N$ denote the binary classification labels of $N$ samples. A *linear classifier* is a function $h_\beta : \mathbb{R}^J \to \{0, 1\}$ defined as $h_\beta(\mathbf{x}) \triangleq \arg\max_{y \in \{0,1\}} \mathbf{x}\beta$, where $\beta = (\beta_1, \ldots, \beta_J) \in \mathbb{R}^J$ is a vector of *parameters* to be estimated. This leads to the following learning problem $\min_{\beta \in \mathbb{R}^J} \lambda R(\boldsymbol{\beta}) + \frac{1}{m} \sum_{i=1}^m L(\boldsymbol{\beta}; x_i, y_i)$, where $L(\boldsymbol{\beta}; x, y)$ is some non-negative, convex *loss*, m is the number of training images, $R(\boldsymbol{\beta})$ is a *regularizer* that avoids overfitting, and $\lambda \in \mathbb{R}$ is the regularization coefficient, whose value can be determined by cross validation.

A common choice of $L$ is the *Log* loss, $L = \log(1/P(y_i|\mathbf{x}_i, \beta))$, where $P(y_i|\mathbf{x}_i, \beta))$ is the *logistic* function $P(y|\mathbf{x}, \beta)) = \frac{1}{Z} \exp(\frac{1}{2} y(\mathbf{x} \cdot \beta))$. This leads to the popular logistic regression (LR) classifier[2]. Structural risk minimization schemes over LR via various forms of regularizations have been widely studied and understood in the literature. In particular, recent asymptotic analysis of the $\ell_1$ norm and $\ell_1/\ell_2$ mixed norm regularized LR proved that under certain conditions the estimated *sparse* coefficient vector $\beta$ enjoys a property called *sparsistency* [34], suggesting their applicability for meaningful variable selection in high-dimensional feature space. In this paper, we employ an LR classifier for our scene classification problem. We investigate content-based compression of the high-dimensional OB representation that exploits raw feature-, object-, and (feature+object)-sparsity, respectively, using LR with appropriate regularization.

**Feature sparsity via $\ell_1$ regularized LR (LR1)**  By letting $R(\boldsymbol{\beta}) \triangleq \|\boldsymbol{\beta}\|_1 = \sum_{j=1}^J |\beta_j|$, we obtain an estimator of $\beta$ that is sparse. The shrinkage function on $\beta$ is applied indistinguishably to all dimensions in the OB representation, and it does not have a mechanism to incorporate any potential coupling of multiple features that are possibly synergistic, e.g., features induced by the same object detector. We call such a sparsity pattern *feature sparsity*, and denote the resultant coefficient estimator by $\beta^{\mathrm{F}}$.

**Object sparsity via $\ell_1/\ell_2$ (group) regularized LR (LRG)**  Recently, a mixed-norm (e.g., $\ell_1/\ell_2$) regularization [36] has been used for recovery of joint sparsity across input dimensions. By letting $R(\boldsymbol{\beta}) \triangleq \|\boldsymbol{\beta}\|_{1,2} = \sum_{j=1}^J \|\boldsymbol{\beta}^j\|_2$, where $\boldsymbol{\beta}^j$ is the $j$-th group (i.e., features grouped by an object $j$), and $\|\cdot\|_2$ is the vector $\ell_2$-norm, we set the feature group to be corresponding to that of all features induced by the same object in the OB. This shrinkage tends to encourage features in the same group to be jointly zero. Therefore, the sparsity is now imposed on object level, rather than merely on raw feature level. Such *structured sparsity* is often desired because it is expected to generate semantically more meaningful lossless compression, that is, out of all the objects in the OB, only a few are needed to represent any given natural image. We call such a sparsity pattern *object sparsity*, and denote the resultant coefficient estimator by $\beta^{\mathrm{O}}$.

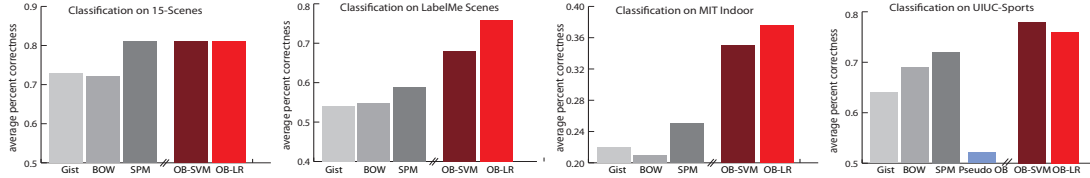

Figure 3: (Best viewed in colors and magnification.) Comparison of classification performance of different features (GIST vs. BOW vs. SPM vs. OB) and classifiers (SVM vs. LR) on (top to down) 15 scene, LabelMe, UIUC-Sports and MIT-Indoor datasets. In the LabelMe dataset, the "ideal" classification accuracy is 90%, where we use the human ground-truth object identities to predict the labels of the scene classes. The blue bar in the last panel is the performance of "pseudo" object bank representation extracted from the same number of "pseudo" object detectors. The values of the parameters in these "pseudo" detectors are generated without altering the original detector structures. In the case of linear classifier, the weights of the classifier are randomly generated from a uniform distribution instead of learned. "Pseudo" OB is then extracted with exactly the same setting as OB.

**Joint object/feature sparsity via $\ell_1/\ell_2 + \ell_1$ (sparse group) regularized LR (LRG1)** The group-regularized LR does not, however, yield sparsity within a group (object) for those groups with non-zero total weights. That is, if a group of parameters is non-zero, they will all be non-zero. Translating to the OB representation, this means there is no scale or spatial location selection for an object. To remedy this, we proposed a composite regularizer, $R(\boldsymbol{\beta}) \triangleq \lambda_1\|\boldsymbol{\beta}\|_{1,2} + \lambda_2\|\boldsymbol{\beta}\|_1$, which conjoin the sparsification effects of both shrinkage functions, and yields sparsity at both the group and individual feature levels. This regularizer necessitates determination of two regularization parameters $\lambda_1$ and $\lambda_2$, and therefore is more difficult to optimize. Furthermore, although the optimization problem for $\ell_1/\ell_2 + \ell_1$ regularized LR is convex, the non-smooth penalty function makes the optimization highly nontrivial. In the Appendix, we derive a coordinate descent algorithm for solving this problem. To conclude, we call the sparse group shrinkage patten *object/feature sparsity*, and denote the resultant coefficient estimator by $\beta^{\mathrm{OF}}$.

# 5 Experiments and Results

**Dataset** We evaluate the OB representation on 4 scene datasets, ranging from generic natural scene images (15-Scene, LabelMe 9-class scene dataset[3]), to cluttered indoor images (MIT Indoor Scene), and to complex event and activity images (UIUC-Sports). Scene classification performance is evaluated by average multi-way classification accuracy over all scene classes in each dataset. We list below the experiment setting for each dataset:

- 15-Scene: This is a dataset of 15 natural scene classes. We use 100 images in each class for training and rest for testing following [19].

- LabelMe: This is a dataset of 9 classes. 50 images randomly drawn images from each scene classes are used for training and 50 for testing.

- MIT Indoor: This is a dataset of 15620 images over 67 indoor scenes assembled by [27]. We follow their experimental setting in [27] by using 80 images from each class for training and 20 for testing.

- UIUC-Sports: This is a dataset of 8 complex event classes. 70 randomly drawn images from each classes are used for training and 60 for testing following [22].

**Experiment Setup** We compare OB in scene classification tasks with different types of conventional image features, such as SIFT-BoW [23, 3], GIST [25] and SPM [19]. An off-the-shelf SVM classifier, and an in-house implementation of the logistic regression (LR) classifier were used on all feature representations being compared. We investigate the behaviors of different structural risk minimization schemes over LR on the OB representation. As introduced in Sec 4, we experimented $\ell_1$ regularized LR (LR1), $\ell_1/\ell_2$ regularized LR (LRG) and $\ell_1/\ell_2 + \ell_1$ regularized LR (LRG1).

## 5.1 Scene Classification

Fig.3 summarizes the results on scene classification based on OB and a set of well known low-level feature representations: GIST [25], Bag of Words (BOW) [3] and Spatial Pyramid Matching

(SPM) [19] on four challenging scene datasets. We show the results of OB using both an LR classifier and a linear SVM [4] We achieve substantially superior performances on three out of four datasets, and are on par with the 15-Scene dataset. The substantial performance gain on the UIUC-Sports and the MIT-Indoor scene datasets illustrates the importance of using a semantically meaningful representation for complex scenes cluttered with objects. For example, the difference between a livingroom and a bedroom is less so in the overall texture (easily captured by BoW or GIST), but more so in the different objects and their arrangements. This result underscores the effectiveness of OB, highlighting the fact that in high-level visual tasks such as complex scene recognition, a higher level image representation can be very useful. We further decompose the spatial structure and semantic meaning encoded in OB by using a "pseudo" OB without semantic meaning. The significant improvement of OB in classification performance over the "pseudo object bank" is largely attributed to the effectiveness of using object detectors trained from image. For each of the existing scene datasets (UIUC-Sports, 15-Scene and MIT-Indoor), we also compare the reported state of the arts performances to our OB algorithm (using a standard LR classifier). This result is shown in Tab.1[5]

## 5.2 Control Experiment: Object Recognition by OB vs. Classemes [33]

OB is constructed from the responses of many objects, which encodes the semantic and spatial information of objects within images. It can be naturally applied to object recognition task. We compare the object recognition performance on the Caltech 256 dataset to [33], a high level image representation obtained as the output of a large number of weakly trained object classifiers on the image. By encoding the spatial locations of the objects within an image, OB (39%) significantly outperforms [33] (36%) on the 256-way classification task, where per-

|  | 15-Scene | UIUC-Sports | MIT-Indoor |
|---|---|---|---|
| state-of -the-art | 72.2%[19] 81.1%[19] | 66.0% [32] 73.4% [22] | 26% [27] |
| OB | 80.9% | 76.3% | 37.6% |

Table 1: Comparison of classification results using OB with reported state-of-the-art algorithms. Many of the algorithms use more complex model and supervised information, whereas our results are obtained by applying simple logistic regression.

formance is measured as the average of the diagonal values of a $256 \times 256$ confusion matrix.

### 5.3 Semantic Feature Sparsification Over OB

In this subsection, we systematically investigate semantic feature sparsification of the OB representation. We focus on the practical issues directly relevant to the effectiveness of OB representation and quality of feature sparsification, and study the following three aspects of the scene classifier: 1) robustness, 2) feasibility of lossless content-based compression, 3) profitability over growing OB.interpretability of predictive features.

#### 5.3.1 Robustness with Respect to Training Sample Size

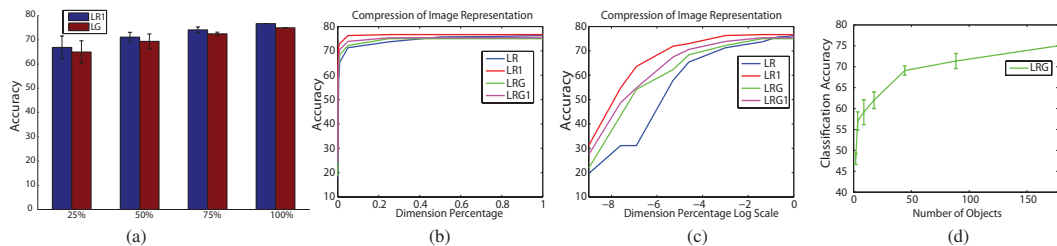

Figure 4: (a) Classification performance (and s.t.d.) w.r.t number of training images. Each pair represents performances of LR1 and LRG respectively. X-axis is the ratio of the training images over the full training dataset (70 images/class). (b) Classification performance w.r.t feature dimension. X-axis is the size of compressed feature dimension, represented as the ratio of the compressed feature dimension over the full OB representation dimension (44604). (c) Same as (b), represented in Log Scale to contrast the performances of different algorithms. (d) Classification performance w.r.t number of object filters. X-axis is the number of object filters. 3 rounds of randomized sampling is performed to choose the object filters from all the object detectors.

The intrinsic high-dimensionness of the OB representation raises a legitimate concern on its demand on training sample size. We investigate the robustness of the logistic regression classifier built on

features selected by LR1 and LRG in this experiment. We train LR1 and LRG on the UIUC-Sports dataset by using multiple sizes of training examples, ranging from 25%, 50%, 75% to 100% of the full training data.

As shown in Fig. 4(a), we observe only moderate drop of performance when the number of training samples decreases from 100% to 25% of the training examples, suggesting that the OB representation is a rich representation where discriminating information residing in a lower dimensional "informative" feature space, which are likely to be retained during feature sparsification, and thereby ensuring robustness under small training data. We explore this issue further in the next experiment.

### 5.3.2 Near Losslessness of Content-based Compression via Regularized Learning

We believe that the OB can offer an *over complete* representation of any natural image. Therefore, there is great room for possibly (near) lossless content-based compression of the image features into a much lower-dimensional, but equally discriminative subspace where key semantic information of the images are preserved, and the quality of inference on images such as scene classification are not compromised significantly. Such compression can be attractive in reducing representation cost of image query, and improving the speed of query inference.

In this experiment, we use the classification performance as a measurement to show how different regularization schemes over LR can preserve the discriminative power. For LR1, LRG and LRG1, cross-validation is used to decide the best regularization parameters. To study the extend of information loss as a function of different number of features being retained in the classifier, we re-train an LR classifier using features from the top $x\%$ percentile of the rank list, where $x$ is a compression scale ranging from 0.05% to 100%. One might think that LR itself when fitted on full input dimensional can also produce a rank list of features for subsequent selection. For comparison purpose, we also include results from the LR-ranked features, as can be seen in Fig.4(b,c), indeed its performance drops faster than all the regularization methods.

In Fig.4 (b), we observe that the classification accuracy drops very slowly as the number of selected features decreases. By excluding 75% feature dimensions, classification performance of each algorithm decreases less than 3%. One point to notice here is that, the non-zero entries only appear in dimensions corresponding to no more than 45 objects for LRG at this point. Even more surprisingly, LR1 and LRG preserve accuracies above 70% when 99% of the feature dimensions are excluded.

Fig. 4 (c) shows more detailed information in the low feature dimension range, which corresponds to a high compression ratio. We observe that algorithms imposing sparsity in features (LR1, LRG, and LRG1) outperform unregularized algorithm (LR) with a larger margin when the compression ratio becomes higher. This reflects that the sparsity learning algorithms are capable of learning the much lower-dimensional, but highly discriminative subspace.

### 5.3.3 Profitability Over Growing OB

We envisage the Object Bank will grow rapidly and constantly as more and more labeled web images become available. This will naturally lead to increasingly richer and higher-dimensional representation of images. We ask, are image inference tasks such as scene classification going to benefit from this trend?

As group regularized LR imposes sparsity on object level, we choose to use it to investigate how the number of objects will affect the discriminative power of OB representation. To simulate what happens when the size of OB grows, we randomly sample subsets of object detectors at 1%, 5%, 10%, 25%, 50% and 75% of total number of objects for multiple rounds. As in Fig.4(d), the classification performance of LRG continuously increases when more objects are incorporated in the OB representation. We conjecture that this is due to the accumulation of discriminative object features, and we believe that future growth of OB will lead to stronger representation power and discriminability of images models build on OB.

### 5.4 Interpretability of the Compressed Representation

Intuitively, a few key objects can discriminate a scene class from another. In this experiment, we aim to discover the *object sparsity* and investigate its interpretability. Again, we use group regularized LR (LRG) since the sparsity is imposed on object level and hence generates a more semantically meaningful compression.

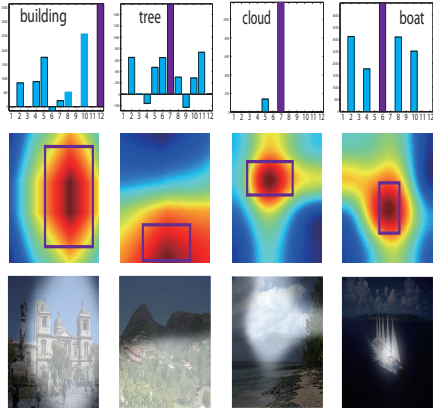

Figure 6: Illustration of the learned $\beta^{OF}$ by LRG1 within an object group. Columns from left to right correspond to "building" in "church" scene, "tree" in "mountain", "cloud" in "beach", and "boat" in "sailing". Top Row: weights of OB dimensions corresponding to different scales, from small to large. The weight of a scale is obtained by summing up the weights of all features corresponding to this scale in $\beta^{OF}$. Middle: Heat map of feature weights in image space at the scale with the highest weight (purple bars above). We project the learned feature weights back to the image by reverting the OB extraction procedure. The purple bounding box shows the size of the object filter at this scale, centered at the peak of the heat map. Bottom: example scene images masked by the feature weights in image space (at the highest weighted scale), highlighting the most relevant object dimension.

We show in Fig.5 the object-wise coefficients of the compression results for 4 sample scene classes. The object weight is obtained by accumulating the coefficient of $\beta^O$ from the feature dimensions of each object (at different scales and spatial locations) learned by LRG. Objects with all zero coefficients in the resultant coefficient estimator are not displayed. Fig.5 shows that objects that are "representative" for each scene are retained by LRG. For example, "sailboat", "boat", and "sky" are objects with very high weight in the "sailing" scene class. This suggests that the representation compression via LRG is virtually based upon the image content and is semantically meaningful; therefore, it is nearly "semantically lossless".

Knowing the important objects learned by the compression algorithm, we further investigate the discriminative

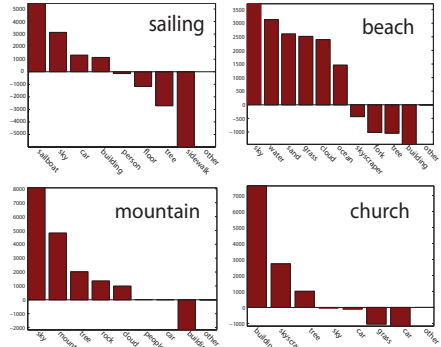

Figure 5: Object-wise coefficients given scene class. Selected objects correspond to non-zero $\beta$ values learned by LRG.

dimensions within the object level. We use LRG1 to examine the learned weights within an object. In Sec.3, we introduce that each feature dimension in the OB representation is directly related to a specific scale, geometric location and object identity. Hence, the weights in $\beta^{OF}$ reflects the importance of an object at a certain scale and location. To verify the hypothesis, we examine the importance of objects across scales by summing up the weights of related spatial locations and pyramid resolutions. We show one representative object in a scene and visualize the feature patterns within the object group. As it is shown in Fig.6(Top), LRG1 has achieved joint object/feature sparsification by zero-out less relevant scales, thus only the most discriminative scales are retained. To analyze how $\beta^{OF}$ reflects the geometric location, we further project the learned coefficient back to the image space by reversing the OB representation extraction procedure. In Fig.6(Middle), we observe that the regions with high intensities are also the locations where the object frequently appears. For example, cloud usually appears in the upper half of a scene in the beach class.

## 6 Conclusion

As we try to tackle higher level visual recognition problems, we show that Object Bank representation is powerful on scene classification tasks because it carries rich semantic level image information. We also apply structured regularization schemes on the OB representation, and achieve nearly lossless semantic-preserving compression. In the future, we will further test OB representation in other useful vision applications, as well as other interesting structural regularization schemes.

**Acknowledgments** L. F-F is partially supported by an NSF CAREER grant (IIS-0845230), a Google research award, and a Microsoft Research Fellowship. E. X is supported by AFOSR FA9550010247, ONR N0001140910758, NSF Career DBI-0546594, NSF IIS- 0713379 and Alfred P. Sloan Fellowship. We thank Wei Yu, Jia Deng, Olga Russakovsky, Bangpeng Yao, Barry Chai, Yongwhan Lim, and anonymous reviewers for helpful comments.

## Footnotes

*indicates equal contributions.

[1]This criterion prevents us from using the Caltech101/256 datasets to train our object detectors [6, 11] where the objects are chosen without any particular considerations of their relevance to daily life pictures.

[2]We choose not to use the popular SVM which correspond to $L$ being a *hinge* loss and $R(\boldsymbol{\beta})$ being a $\ell_2$-regularizer, because under SVM, content-based compression via structured regularization is much harder.

[3]From 100 popular scene names, we obtained 9 classes from the LabelMe dataset in which there are more than 100 images: beach, mountain, bathroom, church, garage, office, sail, street, forest. The maximum number of images in those classes is 1000.

[4]We also evaluate the classification performance of using the detected object location and its detection score of each object detector as the image representation. The classification performance of this representation is 62.0%, 48.3%, 25.1% and 54% on the 15 scene, LabelMe, UIUC-Sports and MIT-Indoor datasets respectively.

[5]We refer to the Appendix for a further discussion of the issue of comparing different algorithms based on different training strategies.

## References

[1] S. Belongie, J. Malik, and J. Puzicha. Shape matching and object recognition using shape contexts. *IEEE PAMI*, pages 509–522, 2002.

[2] L. Bourdev and J. Malik. Poselets: Body Part Detectors Trained Using 3D Human Pose Annotations. *ICCV*, 2009.

[3] G. Csurka, C. Bray, C. Dance, and L. Fan. Visual categorization with bags of keypoints. *Workshop on Statistical Learning in Computer Vision, ECCV*, 2004.

[4] N. Dalal and B. Triggs. Histograms of oriented gradients for human detection. *CVPR*, 2005.

[5] J. Deng, W. Dong, R. Socher, L.-J. Li, K. Li, and L. Fei-Fei. ImageNet: A Large-Scale Hierarchical Image Database. *CVPR*, 2009.

[6] L. Fei-Fei, R. Fergus, and P. Perona. One-Shot learning of object categories. *TPAMI*, 2006.

[7] L. Fei-Fei, R. Fergus, and A. Torralba. Recognizing and learning object categories. Short Course CVPR

[8] A. Farhadi, I. Endres, D. Hoiem and D. Forsyth. Describing objects by their attributes. *CVPR*, 2009.

[9] P. Felzenszwalb, R. Girshick, D. McAllester, and D. Ramanan. Object Detection with Discriminatively Trained Part Based Models. *JAIR*, 29, 2007.

[10] W.T. Freeman and E.H. Adelson. The design and use of steerable filters. *IEEE PAMI*, 1991.

[11] G. Griffin, A. Holub, and P. Perona. Caltech-256 Object Category Dataset. 2007.

[12] A. Hauptmann, R. Yan, W. Lin, M. Christel, and H. Wactlar. Can high-level concepts fill the semantic gap in video retrieval? a case study with broadcast news. *IEEE TMM*, 9(5):958, 2007.

[13] D. Hoiem, A.A. Efros, and M. Hebert. Automatic photo pop-up. *SIGGRAPH 2005*, 24(3):577–584, 2005.

[14] D. Hoiem, A.A. Efros, and M. Hebert. Putting Objects in Perspective. *CVPR*, 2006.

[15] T. Kadir and M. Brady. Scale, saliency and image description. *IJCV*, 45(2):83–105, 2001.

[16] N. Kumar, A. C. Berg, P. N. Belhumeur and S. K. Nayar. Attribute and Simile Classifiers for Face Verification. *ICCV*, 2009.

[17] C.H. Lampert, H. Nickisch and S. Harmeling. Learning to detect unseen object classes by between-class attribute transfer. *CVPR*, 2009.

[18] C.H. Lampert, M.B. Blaschko, T. Hofmann, and S. Zurich. Beyond sliding windows: Object localization by efficient subwindow search. *CVPR*, 2008.

[19] S. Lazebnik, C. Schmid, and J. Ponce. Beyond bags of features: Spatial pyramid matching for recognizing natural scene categories. *CVPR*, 2006.

[20] H.Lee, R.Grosse, R.Ranganath and A. Y. Ng. Convolutional deep belief networks for scalable unsupervised learning of hierarchical representations. *ICML*, 2009.

[21] D.Lewis. Naive (Bayes) at Forty: The Independence Assumption in Information Retrieval. *ECML*, 1998.

[22] L-J. Li and L. Fei-Fei. What, where and who? classifying events by scene and object recognition. *ICCV*, 2007.

[23] D. Lowe. Object recognition from local scale-invariant features. *ICCV*, 1999.

[24] K. Mikolajczyk and C. Schmid. An affine invariant interest point detector. *ECCV*, 2002.

[25] A. Oliva and A. Torralba. Modeling the shape of the scene: a holistic representation of the spatial envelope. *IJCV*, 42, 2001.

[26] P. Perona and J. Malik. Scale-space and edge detection using anisotropic diffusion. *PAMI*, 1990.

[27] A. Quattoni and A. Torralba. Recognizing indoor scenes. *CVPR*, 2009.

[28] A. Rabinovich, A. Vedaldi, C. Galleguillos, E. Wiewiora and S. Belongie. Objects in context. *ICCV*, 2007.

[29] D. Ramanan C. Desai and C. Fowlkes. Discriminative models for multi-class object layout. *ICCV*, 2009.

[30] B.C. Russell, A. Torralba, K.P. Murphy, and W.T. Freeman. Labelme: a database and web-based tool for image annotation. *MIT AI Lab Memo*, 2005.

[31] L. Von Ahn. Games with a purpose. *Computer*, 39(6):92–94, 2006.

[32] C. Wang, D. Blei, and L. Fei-Fei. Simultaneous image classification and annotation. *CVPR*, 2009.

[33] L. Torresani, M. Szummer, and A. Fitzgibbon. Efficient Object Category Recognition Using Classemes. *European Conference of Computer Vision 2010*, pages 776–789, 2010.

[34] P.Ravikumar, M.Wainwright, J.Lafferty. High-Dimensional Ising Model Selection Using L1-Regularized Logistic Regression. *Annals of Statistics*, 2009.

[35] J. Vogel and B. Schiele. Semantic modeling of natural scenes for content-based image retrieval. *International Journal of Computer Vision*, 2007.

[36] M. Yuan and Y. Lin. Model selection and estimation in regression with grouped variables. *Journal of the Royal Statistical Society: Series B (Statistical Methodology)*, 2006.

